# Tight Bounds for the VC-Dimension of Piecewise Polynomial Networks

**Akito Sakurai**
School of Knowledge Science
Japan Advanced Institute of Science and Technology
Nomi-gun, Ishikawa 923-1211, Japan.
CREST, Japan Science and Technology Corporation.
*ASakurai@jaist.ac.jp*

## Abstract

$O(ws(s \log d + \log(dqh/s)))$ and $O(ws((h/s) \log q) + \log(dqh/s))$ are upper bounds for the VC-dimension of a set of neural networks of units with piecewise polynomial activation functions, where $s$ is the depth of the network, $h$ is the number of hidden units, $w$ is the number of adjustable parameters, $q$ is the maximum of the number of polynomial segments of the activation function, and $d$ is the maximum degree of the polynomials; also $\Omega(ws \log(dqh/s))$ is a lower bound for the VC-dimension of such a network set, which are tight for the cases $s = \Theta(h)$ and $s$ is constant. For the special case $q = 1$, the VC-dimension is $\Theta(ws \log d)$.

## 1 Introduction

In spite of its importance, we had been unable to obtain VC-dimension values for practical types of networks, until fairly tight upper and lower bounds were obtained ([6], [8], [9], and [10]) for linear threshold element networks in which all elements perform a threshold function on weighted sum of inputs. Roughly, the lower bound for the networks is $(1/2)w \log h$ and the upper bound is $w \log h$ where $h$ is the number of hidden elements and $w$ is the number of connecting weights (for one-hidden-layer case $w \approx nh$ where $n$ is the input dimension of the network).

In many applications, though, sigmoidal functions, specifically a typical sigmoid function $1/(1 + \exp(-x))$, or piecewise linear functions for economy of calculation, are used instead of the threshold function. This is mainly because the differentiability of the functions is needed to perform backpropagation or other learning algorithms. Unfortunately explicit bounds obtained so far for the VC-dimension of sigmoidal networks exhibit large gaps ($O(w^2h^2)$ ([3]), $\Omega(w \log h)$ for bounded depth

and $\Omega(wh)$ for unbounded depth) and are hard to improve. For the piecewise linear case, Maass obtained a result that the VC-dimension is $O(w^2 \log q)$, where $q$ is the number of linear pieces of the function ([5]).

Recently Koiran and Sontag ([4]) proved a lower bound $\Omega(w^2)$ for the piecewise polynomial case and they claimed that an open problem that Maass posed if there is a matching $w^2$ lower bound for the type of networks is solved. But we still have something to do, since they showed it only for the case $w = \Theta(h)$ and the number of hidden layers being unbounded; also $O(w^2)$ bound has room to improve.

We in this paper improve the bounds obtained by Maass, Koiran and Sontag and consequently show the role of polynomials, which can not be played by linear functions, and the role of the constant functions that could appear for piecewise polynomial case, which cannot be played by polynomial functions.

After submission of the draft, we found that Bartlett, Maiorov, and Meir had obtained similar results prior to ours (also in this proceedings). Our advantage is that we clarified the role played by the degree and number of segments concerning the both bounds.

## 2    Terminology and Notation

log stands for the logarithm base 2 throughout the paper.

The *depth* of a network is the length of the longest path from its external inputs to its external output, where the length is the number of units on the path. Likewise we can assign a *depth* to each unit in a network as the length of the longest path from the external input to the output of the unit. A *hidden layer* is a set of units at the same depth other than the depth of the network. Therefore a depth $L$ network has $L - 1$ hidden layers.

In many cases $\mathbf{w}$ will stand for a vector composed of all the connection weights in the network (including threshold values for the threshold units) and $w$ is the length of $\mathbf{w}$. The number of units in the network, excluding "input units," will be denoted by $h$; in other words, the number of hidden units plus one, or sometimes just the number of hidden units. A function whose range is $\{0, 1\}$ (a set of 0 and 1) is called a *Boolean-valued function*.

## 3    Upper Bounds

To obtain upper bounds for the VC-dimension we use a *region counting argument*, developed by Goldberg and Jerrum [2]. The VC-dimension of the network, that is, the VC-dimension of the function set $\{f_G(\mathbf{w}; \cdot) \mid \mathbf{w} \in \mathcal{R}^w\}$ is upper bounded by

$$\max\left\{ N \mid 2^N \leq \max_{\mathbf{x}_1,\ldots,\mathbf{x}_N} N_{cc}\left(\mathcal{R}^w - \bigcup_{i=1}^{N}\mathcal{N}(f_G(\mathbf{w}; \mathbf{x}_i))\right)\right\} \qquad (3.1)$$

where $N_{cc}(\cdot)$ is the number of connected components and $\mathcal{N}(f)$ is the set $\{\mathbf{w} \mid f(\mathbf{w}) = 0\}$.

The following two theorems are convenient. Refer [11] and [7] for the first theorem. The lemma followed is easily proven.

**Theorem 3.1.** *Let $f_G(\mathbf{w}; \mathbf{x}_i)$ $(1 \leq i \leq N)$ be real polynomials in $\mathbf{w}$, each of degree $d$ or less. The number of connected components of the set $\bigcap_{i=1}^{m}\{\mathbf{w} \mid f_G(\mathbf{w}; \mathbf{x}_i) = 0\}$ is bounded from above by $2(2d)^w$ where $w$ is the length of $\mathbf{w}$.*

**Lemma 3.2.** *If* $m \geq w(\log C + \log \log C + 1)$, *then* $2^m > (mC/w)^w$ *for* $C \geq 4$.

First let us consider the polynomial activation function case.

**Theorem 3.3.** *Suppose that the activation function are polynomials of degree at most $d$. $O(ws \log d)$ is an upper bound of the VC-dimension for the networks with depth $s$. When $s = \Theta(h)$ the bound is $O(wh \log d)$. More precisely $ws(\log d + \log \log d + 2)$ is an upper bound. Note that if we allow a polynomial as the input function, $d_1 d_2$ will replace $d$ above where $d_1$ is the maximum degree of the input functions and $d_2$ is that of the activation functions.*

The theorem is clear from the facts that the network function ($f_G$ in (3.1)) is a polynomial of degree at most $d^s + d^{s-1} + \cdots + d$, Theorem 3.1 and Lemma 3.2.

For the piecewise linear case, we have two types of bounds. The first one is suitable for bounded depth cases (*i.e.* the depth $s = o(h)$) and the second one for the unbounded depth case (*i.e.* $s = \Theta(h)$).

**Theorem 3.4.** *Suppose that the activation functions are piecewise polynomials with at most $q$ segments of polynomials degree at most $d$. $O(ws(s \log d + \log(dqh/s)))$ and $O(ws((h/s) \log q) + \log(dqh/s))$ are upper bounds for the VC-dimension, where $s$ is the depth of the network. More precisely, $ws((s/2) \log d + \log(qh))$ and $ws((h/s) \log q + \log d)$ are asymptotic upper bounds. Note that if we allow a polynomial as the input function then $d_1 d_2$ will replace $d$ above where $d_1$ is the maximum degree of the input functions and $d_2$ is that of the activation functions.*

*Proof.* We have two different ways to calculate the bounds. First

$$N_{cc}\left(\mathcal{R}^w - \bigcup_{i=1}^N \mathcal{N}(f_G(\mathbf{w}; \mathbf{x}_i))\right)$$
$$\leq \prod_{j=1}^s \max N_{cc}\left(\mathcal{R}^{w_1+\cdots+w_j} - \bigcup \mathcal{N}(f_{G_1}(\mathbf{w}_1 \circ \cdots \circ \mathbf{w}_j; \mathbf{x}_i))\right)$$
$$\leq \prod_{j=1}^s \left(\frac{8eNqh_s(d^{j-1}+\cdots+d+1)d}{w_1+\cdots+w_j}\right)^{w_1+\cdots+w_j}$$
$$\leq \left(\frac{8eNqd^{(s+3)/2}(h/s)}{w}\right)^{ws}.$$

where $h_i$ is the number of hidden units in the $i$-th layer and $\circ$ is an operator to form a new vector by concatenating the two. From this we get an asymptotic upper bound $ws((s/2) \log d + \log(qh))$ for the VC-dimension.

Secondly

$$N_{cc}\left(\mathcal{R}^w - \bigcup_{i=1}^N \mathcal{N}(f_G(\mathbf{w}; \mathbf{x}_i))\right) \leq N_{cc}\left(\mathcal{R}^w - \bigcup_{i=1}^N \bigcup_{j=1}^{q^h} \mathcal{N}(f_{G,j}(\mathbf{w}; \mathbf{x}_i))\right) \leq \left(\frac{8eNq^h d^s}{w}\right)^w$$

From this we get an asymptotic upper bound $ws((h/s) \log q + \log d)$ for the VC-dimension. Combining these two bounds we get the result. Note that $s$ in $\log(dqh/s)$ in it is introduced to eliminate unduly large term emerging when $s = \Theta(h)$. ☐

## 4 Lower Bounds for Polynomial Networks

**Theorem 4.1** *Let us consider the case that the activation function are polynomials of degree at most $d$. $\Omega(ws \log d)$ is a lower bound of the VC-dimension for the networks with depth $s$. When $s = \Theta(h)$ the bound is $\Omega(wh \log d)$, More precisely,*

*$(1/16)w(s-6)\log d$ is an asymptotic lower bound where $d$ is the degree of activation functions and is a power of two and $h$ is restricted to $O(n^2)$ for input dimension $n$.*

The proof consists of several lemmas. The network we are constructing will have two parts: an encoder and a decoder. We deliberately fix the $N$ input points. The decoder part has fixed underlying architecture but also fixed connecting weights whereas the encoder part has variable weights so that for any given binary outputs for the input points the decoder could output the specified value from the codes in which the output value is encoded by the encoder.

First we consider the decoder, which has two real inputs and one real output. One of the two inputs $y$ holds a code of a binary sequence $b_1, b_2, \ldots, b_m$ and the other $x$ holds a code of a binary sequence $c_1, c_2, \ldots, c_m$. The elements of the latter sequence are all 0's except for $c_j = 1$, where $c_j = 1$ orders the decoder to output $b_j$ from it and consequently from the network.

We show two types of networks; one of which has activation functions of degree at most two and has the VC-dimension $w(s-1)$ and the other has activation functions of degree $d$ a power of two and has the VC-dimension $w(s-5)\log d$.

We use for convenience two functions $\mathcal{H}_\theta(x) = 1$ if $x \geq \theta$ and 0 otherwise and $\mathcal{H}_{\theta,\phi}(x) = 1$ if $x \geq \phi$, 0 if $x \leq \theta$, and undefined otherwise. Throughout this section we will use a simple logistic function $\rho(x) = (16/3)x(1-x)$ which has the following property.

**Lemma 4.2.** *For any binary sequence $b_1, b_2, \ldots, b_m$, there exists an interval $[x_1, x_2]$ such that $b_i = \mathcal{H}_{1/4,3/4}(\rho^i(x))$ and $0 \leq \rho^i(x) \leq 1$ for any $x \in [x_1, x_2]$.*

The next lemmas are easily proven.

**Lemma 4.3.** *For any binary sequence $c_1, c_2, \ldots, c_m$ which are all 0's except for $c_j = 1$, there exists $x_0$ such that $c_i = \mathcal{H}_{1/4,3/4}(\rho^i(x_0))$. Specifically we will take $x_0 = \rho_L^{-(j-1)}(1/4)$, where $\rho_L^{-1}(x)$ is the inverse of $\rho(x)$ on $[0, 1/2]$. Then $\rho^{j-1}(x_0) = 1/4$, $\rho^j(x_0) = 1$, $\rho^i(x_0) = 0$ for all $i > j$, and $\rho^{j-i}(x_0) \leq (1/4)^i$ for all positive $i \leq j$.*

*Proof.* Clear from the fact that $\rho(x) \geq 4x$ on $[0, 1/4]$.  □

**Lemma 4.4.** *For any binary sequence $b_1, b_2, \ldots, b_m$, take $y$ such that $b_i = \mathcal{H}_{1/4,3/4}(\rho^i(y))$ and $0 \leq \rho^i(y) \leq 1$ for all $i$ and $x_0 = \rho_L^{-(j-1)}(1/4)$, then $\mathcal{H}_{7/12,3/4}\left(\sum_{i=1}^m \rho^i(x_0)\rho^i(y)\right) = b_j$, i.e. $\mathcal{H}_0\left(\sum_{i=1}^m \rho^i(x_0)\rho^i(y) - 2/3\right) = b_j$.*

*Proof.* If $b_j = 0$, $\sum_{i=1}^m \rho^i(x_0)\rho^i(y) = \sum_{i=1}^j \rho^i(x_0)\rho^i(y) \leq \rho^j(y) + \sum_{i=1}^{j-1}(1/4)^i < \rho^j(y) + (1/3) \leq 7/12$. If $b_j = 1$, $\sum_{i=1}^m \rho^i(x_0)\rho^i(y) > \rho^j(x_0)\rho^j(y) \geq 3/4$.  □

By the above lemmas, the network in Figure 1 (left) has the following function:

  · Suppose that a binary sequence $b_1, \ldots, b_m$ and an integer $j$ is given. Then we can present $y$ that depends only on $b_1, \ldots, b_m$ and $x_0$ that depends only on $j$ such that $b_j$ is output from the decoder.

Note that we use $(x+y)^2 - (x-y)^2 = 4xy$ to realize a multiplication unit.

For the case of degree of higher than two we have to construct a bit more complicated one by using another simple logistic function $\mu(x) = (36/5)x(1-x)$. We need the next lemma.

**Lemma 4.5.** *Take $x_0 = \mu_L^{-(j-1)}(1/6)$, where $\mu_L^{-1}(x)$ is the inverse of $\mu(x)$ on $[0, 1/2]$. Then $\mu^{j-1}(x_0) = 1/6$, $\mu^j(x_0) = 1$, $\mu^i(x_0) = 0$ for all $i > j$, and $\mu^{j-i}(x_0) =$*

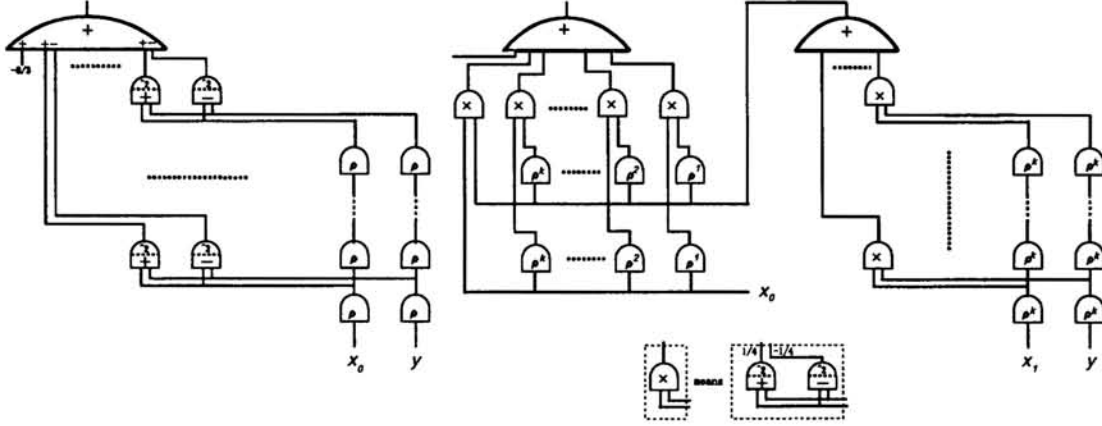

Figure 1: Network architecture consisting of polynomials of order two (left) and those of order of power of two (right).

$(1/6)^i$ *for all* $i > 0$ *and* $\leq j$.

*Proof.* Clear from the fact that $\mu(x) \geq 6x$ on $[0, 1/6]$. ☐

**Lemma 4.6.** *For any binary sequence* $b_1, b_2, \ldots, b_k,\ b_{k+1}, b_{k+2}, \ldots, b_{2k},$ $\ldots, b_{(m-1)k+1}, \ldots, b_{mk}$ *take* $y$ *such that* $b_i = \mathcal{H}_{1/4, 3/4}(\rho^i(y))$ *and* $0 \leq \rho^i(y) \leq 1$ *for all* $i$. *Moreover for any* $1 \leq j \leq m$ *and any* $1 \leq l \leq k$ *take* $x_1 = \mu_L^{-(j-1)}(1/6)$, *and* $x_0 = \mu_L^{-(l-1)}(1/6^k)$. *Then for* $z = \sum_{i=1}^{m} \rho^{ik}(y) \mu^{ik}(x_1)$, $\mathcal{H}_0\left(\sum_{i=0}^{k-1} \rho^i(z)\mu^i(x_0) - (1/2)\right) = b_{kj+l}$ *holds.*

**Lemma 4.7.** *If* $0 < \rho^i(x) < 1$ *for any* $0 < i \leq l$, *take an* $\epsilon$ *such that* $(16/3)^l \epsilon < 1/4$. *Then* $\rho^l(x) - (16/3)^l \epsilon < \rho^l(x + \epsilon) < \rho^l(x) + (16/3)^l \epsilon$.

*Proof.* There are four cases depending on whether $\rho^{l-1}(x + \epsilon)$ is on the uphill or downhill of $\rho$ and whether $x$ is on the uphill or downhill of $\rho^{l-1}$. The proofs are done by induction.

First suppose that the two are on the uphill. Then $\rho^l(x + \epsilon) = \rho(\rho^{l-1}(x + \epsilon)) < \rho(\rho^{l-1}(x) + (16/3)^{l-1}\epsilon)) < \rho^l(x) + (16/3)^l \epsilon$. Secondly suppose that $\rho^{l-1}(x + \epsilon)$ is on the uphill but $x$ is on the downhill. Then $\rho^l(x + \epsilon) = \rho(\rho^{l-1}(x + \epsilon)) > \rho(\rho^{l-1}(x) - (16/3)^{l-1}\epsilon)) > \rho^l(x) - (16/3)^l \epsilon$. The other two cases are similar. ☐

*Proof of Lemma 4.6.* We will show that the difference between $\rho^{jk+l}(y)$ and $\sum_{i=0}^{k-1} \rho^i(z)\mu^i(x_0)$ is sufficiently small. Clearly $z = \sum_{i=1}^{m} \mu^{ik}(x_1)\rho^{ik}(y) = \sum_{i=1}^{j} \mu^{ik}(x_1)\rho^{ik}(y) \leq \rho^{jk}(y) + \sum_{i=1}^{j-1}(1/6^k)^i < \rho^{jk}(y) + 1/(6^k - 1)$ and $\rho^{jk}(y) < z$. If $z$ is on the uphill of $\rho^l$ then by using the above lemma, we get $\sum_{i=0}^{k-1} \rho^i(z)\mu^i(x_0) = \sum_{i=0}^{l} \rho^i(z)\mu^i(x_0) < \rho^l(z) + 1/(6^k - 1) < \rho^{jk+l}(y) + (1 + (16/3)^l)(1/(6^k - 1)) < \rho^{jk+l}(y) + 1/4$ (note that $l \leq k - 1$ and $k \geq 2$). If $z$ is on the downhill of $\rho^l$ then by using the above lemma, we get $\sum_{i=0}^{k-1} \rho^i(z)\mu^i(x_0) = \sum_{i=0}^{l} \rho^i(z)\mu^i(x_0) > \rho^l(z) > \rho^l(\rho^{jk}(y)) - (16/3)^l(1/(6^k - 1)) > \rho^{jk+l}(y) - 1/4$. ☐

Next we show the encoding scheme we adopted. We show only the case $w = \Theta(h^2)$ since the case $w = \Theta(h)$ or more generally $w = O(h^2)$ is easily obtained from this.

**Theorem 4.8** *There is a network of $2n$ inputs, $2h$ hidden units with $h^2$ weights* **w**,

and $h^2$ sets of input values $\mathbf{x}_1, \ldots, \mathbf{x}_{h^2}$ such that for any set of values $y_1, \ldots, y_{h^2}$ we can chose $\mathbf{w}$ to satisfy $y_i = f_G(\mathbf{w}; \mathbf{x}_i)$.

*Proof.* We extensively utilize the fact that monomials obtained by choosing at most $k$ variables from $n$ variables with repetition allowed (say $x_1^2 x_2 x_6$) are all linearly independent ([1]). Note that the number of monomials thus formed is $\binom{n+m}{m}$.

Suppose for simplicity that we have $2n$ inputs and $2h$ main hidden units (we have other hidden units too), and $h = \binom{n+m}{m}$. By using multiplication units (in fact each is a composite of two squaring units and the outputs are supposed to be summed up as in Figure 1), we can form $h = \binom{n+m}{m}$ linearly independent monomials composed of variables $x_1, \ldots, x_n$ by using at most $(m-1)h$ multiplication units (or $h$ nominal units when $m = 1$). In the same way, we can form $h$ linearly independent monomials composed of variables $x_{n+1}, \ldots, x_{2n}$. Let us denote the monomials by $u_1, \ldots, u_h$ and $v_1, \ldots, v_h$.

We form a subnetwork to calculate $\sum_{j=1}^{h}(\sum_{i=1}^{h} w_{i,j} u_i) v_j$ by using $h$ multiplication units. Clearly the calculated result $y$ is the weighted sum of monomials described above where the weights are $w_{i,j}$ for $1 \leq i, j \leq h$.

Since $y = f_G(\mathbf{w}; \mathbf{x})$ is a linear combination of linearly independent terms, if we choose appropriately $h^2$ sets of values $\mathbf{x}_1, \ldots, \mathbf{x}_{h^2}$ for $\mathbf{x} = (x_1, \ldots, x_{2n})$, then for any assignment of $h^2$ values $y_1, \ldots, y_{h^2}$ to $y$ we have a set of weights $\mathbf{w}$ such that $y_i = f(\mathbf{x}_i, \mathbf{w})$.                                                                                    □

*Proof of Theorem 4.1.* The whole network consists of the decoder and the encoder. The input points are the Cartesian product of the above $\mathbf{x}_1, \ldots, \mathbf{x}_{h^2}$ and $\{x_0$ defined in Lemma 4.4 for $b_j = 1 \mid 1 \leq j \leq s'\}$ for some $h$ where $s'$ is the number of bits to be encoded. This means that we have $h^2 s$ points that can be shattered.

Let the number of hidden layers of the decoder be $s$. The number of units used for the decoder is $4(s-1)+1$ (for the degree 2 case which can decode at most $s$ bits) or $4(s-3)+4(k-1)+1$ (for the degree $2^k$ case which can decode at most $(s-2)k$ bits). The number of units used for the encoder is less than $4h$; we though have constraints on $s$ (which dominates the depth of the network) and $h$ (which dominates the number of units in the network) that $h \leq \binom{n+m}{m}$ and $m = O(s)$ or roughly $\log h = O(s)$ be satisfied.

Let us chose $m = 2$ ($m = \log s$ is a better choise). As a result, by using $4h + 4(s-1)+1$ (or $4h + 4(s-3)+4(k-1)+1$) units in $s+2$ layers, we can shatter $h^2 s$ (or $h^2(s-2)\log d$) points; or asymptotically by using $h$ units $s$ layers we can shatter $(1/16)w(s-3)$ (or $(1/16)w(s-5)\log d$) points.                                                                                    □

## 5   Piecewise Polynomial Case

**Theorem 5.1.** *Let us consider a set of networks of units with linear input functions and piecewise polynomial (with $q$ polynomial segments) activation functions. $\Omega(ws \log(dqh/s))$ is a lower bound of the VC-dimension, where $s$ is the depth of the network and $d$ is the maximum degree of the activation functions. More precisely, $(1/16)w(s-6)(\log d + \log(h/s) + \log q)$ is an asymptotic lower bound.*

For the scarcity of space, we give just an outline of the proof. Our proof is based on that of the polynomial networks. We will use $h$ units with activation function of $q \geq 2$ polynomial segments of degree at most $d$ in place of each of $\rho^k$ unit in the decoder, which give the ability of decoding $\log dqh$ bits in one layer and $s \log dqh$ bits in total by $\Theta(sh)$ units in total. If $h$ designates the total number of units, the

number of the decodable bits is represented as $\log(dqh/s)$.

In the following for simplicity we suppose that $dqh$ is a power of 2. Let $\rho^k(x)$ be the $k$ composition of $\rho(x)$ as usual *i.e.* $\rho^k(x) = \rho(\rho^{k-1}(x))$ and $\rho^1(x) = \rho(x)$. Let $\rho^{\log d,l}(x) = \rho^{\log d}(\lambda^l(x))$, where $\lambda(x) = 4x$ if $x \le 1/2$ and $4 - 4x$ otherwise, which by the way has $2^l$ polynomial segments.

Now the $\rho^k$ unit in the polynomial case is replaced by the array $\rho^{\log d,\log q,\log h}(x)$ of $h$ units that is defined as follows:

(i) $\rho^{\log d,\log q,1}(x)$ is an array of two units; one is $\rho^{\log d,\log q}(\lambda^+(x))$ where $\lambda^+(x) = 4x$ if $x \le 1/2$ and 0 otherwise and the other is $\rho^{\log d,\log q}(\lambda^-(x))$ where $\lambda^-(x) = 0$ if $x \le 1/2$ and $4 - 4x$ otherwise.

(ii) $\rho^{\log d,\log q,m}(x)$ is the array of $2^m$ units, each with one of the functions $\rho^{\log d,\log q}(\lambda^{\pm}(\cdots(\lambda^{\pm}(x))\cdots))$ where $\lambda^{\pm}(\cdots(\lambda^{\pm}(x))\cdots)$ is the $m$ composition of $\lambda^+(x)$ or $\lambda^-(x)$. Note that $\lambda^{\pm}(\cdots(\lambda^{\pm}(x))\cdots)$ has at most three linear segments (one is linear and the others are constant 0) and the sum of $2^m$ possible combinations $f(\lambda^{\pm}(\cdots(\lambda^{\pm}(x))\cdots))$ is equal to $f(\lambda^m(x))$ for any function $f$ such that $f(0) = 0$.

Then lemmas similar to the ones in the polynomial case follow.

## References

[1] Anthony, M: Classification by polynomial surfaces, *NeuroCOLT Technical Report Series*, NC-TR-95-011 (1995).

[2] Goldberg, P. and M. Jerrum: Bounding the Vapnik-Chervonenkis dimension of concept classes parameterized by real numbers, *Proc. Sixth Annual ACM Conference on Computational Learning Theory*, 361–369 (1993).

[3] Karpinski, M. and A. Macintyre, Polynomial bounds for VC dimension of sigmoidal neural networks, *Proc. 27th ACM Symposium on Theory of Computing*, 200–208 (1995).

[4] Koiran, P. and E. D. Sontag: Neural networks with quadratic VC dimension, *Journ. Comp. Syst. Sci.*, **54**, 190–198(1997).

[5] Maass, W. G.: Bounds for the computational power and learning complexity of analog neural nets, *Proc. 25th Annual Symposium of the Theory of Computing*, 335-344 (1993).

[6] Maass, W. G.: Neural nets with superlinear VC-dimension, *Neural Computation*, **6**, 877–884 (1994)

[7] Milnor, J.: On the Betti numbers of real varieties, *Proc. of the AMS*, **15**, 275–280 (1964).

[8] Sakurai, A.: Tighter Bounds of the VC-Dimension of Three-layer Networks, *Proc. WCNN'93*, III, 540–543 (1993).

[9] Sakurai, A.: On the VC-dimension of depth four threshold circuits and the complexity of Boolean-valued functions, *Proc. ALT93 (LNAI 744)*, 251–264 (1993); refined version is in *Theoretical Computer Science*, **137**, 109-127 (1995).

[10] Sakurai, A.: On the VC-dimension of neural networks with a large number of hidden layers, *Proc. NOLTA'93*, IEICE, 239–242 (1993).

[11] Warren, H. E.: Lower bounds for approximation by nonlinear manifolds, *Trans. AMS*, **133**, 167–178, (1968).
